# Fitted Q-iteration by Advantage Weighted Regression

**Gerhard Neumann**
Institute for Theoretical Computer Science
Graz University of Technology
A-8010 Graz, Austria
gerhard@igi.tu-graz.ac.at

**Jan Peters**
Max Planck Institute for Biological Cybernetics
D-72076 Tübingen, Germany
mail@jan-peters.net

## Abstract

Recently, fitted Q-iteration (FQI) based methods have become more popular due to their increased sample efficiency, a more stable learning process and the higher quality of the resulting policy. However, these methods remain hard to use for continuous action spaces which frequently occur in real-world tasks, e.g., in robotics and other technical applications. The greedy action selection commonly used for the policy improvement step is particularly problematic as it is expensive for continuous actions, can cause an unstable learning process, introduces an optimization bias and results in highly non-smooth policies unsuitable for real-world systems. In this paper, we show that by using a soft-greedy action selection the policy improvement step used in FQI can be simplified to an inexpensive advantage-weighted regression. With this result, we are able to derive a new, computationally efficient FQI algorithm which can even deal with high dimensional action spaces.

## 1 Introduction

Reinforcement Learning [1] addresses the problem of how autonomous agents can improve their behavior using their experience. At each time step $t$ the agent can observe its current state $s_t \in \mathcal{X}$ and chooses an appropriate action $a_t \in \mathcal{A}$. Subsequently, the agent gets feedback on the quality of the action, i.e., the reward $r_t = r(s_t, a_t)$, and observes the next state $s_{t+1}$. The goal of the agent is to maximize the accumulated reward expected in the future. In this paper, we focus on learning policies for continuous, multi-dimensional control problems. Thus the state space $\mathcal{X}$ and action space $\mathcal{A}$ are continuous and multi-dimensional, meaning that discretizations start to become prohibitively expensive.

While discrete-state/action reinforcement learning is a widely studied problem with rigorous convergence proofs, the same does not hold true for continuous states and actions. For continuous state spaces, few convergence guarantees exist and pathological cases of bad performance can be generated easily [2]. Moreover, many methods cannot be transferred straightforwardly to continuous actions.

Current approaches often circumvent continuous action spaces by focusing on problems where the actor can rely on a discrete set of actions, e.g., when learning a policy for driving to a goal in minimum time, an actor only needs three actions: the maximum acceleration when starting, zero acceleration at maximum velocity and maximum throttle down when the goal is sufficiently close for a point landing. While this approach (called bang-bang in traditional control) works for the large class of minimum time control problems, it is also a limited approach as cost functions relevant to the real-world incorporate much more complex constraints, e.g., cost-functions in biological systems often punish the jerkiness of the movement [3], the amount of used metabolic energy [4] or the variance at the end-point [5]. For physical technical systems, the incorporation of further optimization criteria is of essential importance; just as a minimum time policy is prone to damage the car on the long-run, a similar policy would be highly dangerous for a robot and its environment

and the resulting energy-consumption would reduce its autonomy. More complex, action-dependent immediate reward functions require that much larger sets of actions are being employed.

We consider the use of continuous actions for fitted Q-iteration (FQI) based algorithms. FQI is a batch mode reinforcement learning (BMRL) algorithm. The algorithm mantains an estimate of the state-action value function $Q(\mathbf{s}, \mathbf{a})$ and uses the greedy operator $\max_a Q(\mathbf{s}, \mathbf{a})$ on the action space for improving the policy. While this works well for discrete action spaces, the greedy operation is hard to perform for high-dimensional continuous actions. For this reason, the application of fitted Q-iteration based methods is often restricted to low-dimensional action spaces which can be efficiently discretized. In this paper, we show that the use of a stochastic soft-max policy instead of a greedy policy allows us to reduce the policy improvement step used in FQI to a simple advantage-weighted regression. The greedy operation $\max_a Q(\mathbf{s}, \mathbf{a})$ over the actions is replaced by a less harmful greedy operation over the parameter space of the value function. This result allows us to derive a new, computationally efficient algorithm which is based on Locally-Advantage-WEighted Regression (LAWER).

We test our algorithm on three different benchmark tasks, i.e., the pendulum swing-up [6], the acrobot swing-up [1] and a dynamic version of the puddle-world [7] with 2 and 3 dimensions. We show that in spite of the soft-greedy action selection, our algorithm is able to produce high quality policies.

## 2   Fitted Q-Iteration

In fitted Q-iteration [8, 6, 9] (FQI), we assume that all the experience of the agent up to the current time is given in the form $H = \{< \mathbf{s}_i, \mathbf{a}_i, r_i, \mathbf{s}'_i >\}_{1 \leq i \leq N}$. The task of the learning algorithm is to estimate an optimal control policy from this historical data. FQI approximates the state-action value function $Q(\mathbf{s}, \mathbf{a})$ by iteratively using supervised regression techniques. New target values for the regression are generated by

$$\tilde{Q}_{k+1}(i) \;=\; r_i + \gamma V_k(\mathbf{s}'_i) = r_i + \gamma \max_{\mathbf{a}'} Q_k(\mathbf{s}'_i, \mathbf{a}'). \qquad (1)$$

The regression problem for finding the function $Q_{k+1}$ is defined by the list of data-point pairs $D_k$ and the regression procedure Regress

$$D_k(Q_k) = \left\{ \left[ (\mathbf{s}_i, \mathbf{a}_i), \tilde{Q}_{k+1}(i) \right]_{1 \leq i \leq N} \right\}, \quad Q_{k+1} = \mathrm{Regress}(D_k(Q_k)) \qquad (2)$$

FQI can be viewed as approximate value iteration with state-action value functions [9]. Previous experiments show that function approximators such as neural networks [6], radial basis function networks [8], CMAC [10] and regression trees [8] can be employed in this context. In [9], performance bounds for the value function approximation are given for a wide range of function approximators. The performance bounds also hold true for continuous action spaces, but only in the case of an actor-critic variant of FQI. Unfortunately, to our knowledge, no experiments with this variant exist in the literature. Additionally, it is not clear how to apply this actor-critic variant efficiently for nonparametric function approximators.

FQI has proven to outperform classical online RL methods in many applications [8]. Nevertheless, FQI relies on the greedy action selection in Equation (1). Thus, the algorithm frequently requires a discrete set of actions and generalization to continuous actions is not straightforward. Using the greedy operator for continuous action spaces is a hard problem by itself as the use of expensive optimization methods is needed for high dimensional actions. Moreover the returned values of the greedy operator often result in an optimization bias causing an unstable learning process, including oscillations and divergence [11]. For a comparison with our algorithm, we use the Cross-Entropy (CE) optimization method [12] to find the maximum Q-values. In our implementation, we maintain a Gaussian distribution for the belief of the optimal action. We sample $n_{CE}$ actions from this distribution. Then, the best $e_{CE} < n_{CE}$ actions (with the highest Q-values) are used to update the parameters of this distribution. The whole process is repeated for $k_{CE}$ iterations, starting with a uniformly distributed set of sample actions.

FQI is inherently an offline method - given historical data, the algorithm estimates the optimal policy. However, FQI can also be used for online learning. After the FQI algorithm is finished, new episodes can be collected with the currently best inferred policy and the FQI algorithm is restarted.

# 3 Fitted Q-Iteration by Advantage Weighted Regression

A different method for policy updates in continuous action spaces is reinforcement learning by reward-weighted regression [13]. As shown by the authors, the action selection problem in the immediate reward RL setting with continuous actions can be formulated as expectation-maximization (EM) based algorithm and, subsequently, reduced to a reward-weighted regression. The weighted regression can be applied with ease to high-dimensional action spaces; no greedy operation in the action space is needed. While we do not directly follow the work in [13], we follow the general idea.

## 3.1 Weighted regression for value estimation

In this section we consider the task of estimating the value function $V$ of a stochastic policy $\pi(\cdot|\mathbf{s})$ when the state-action value function $Q$ is given. The value function can be calculated by $V(\mathbf{s}) = \int_{\mathbf{a}} \pi(\mathbf{a}|\mathbf{s})Q(\mathbf{s},\mathbf{a})d\mathbf{a}$. Yet, the integral over the action space is hard to perform for continuous actions. However, we will show how we can approximate the value function without the evaluation of this integral. Consider the quadratic error function

$$\text{Error}(\hat{V}) = \int_{\mathbf{s}} \mu(\mathbf{s}) \left( \int_{\mathbf{a}} \pi(\mathbf{a}|\mathbf{s})Q(\mathbf{s},\mathbf{a})d\mathbf{a} - \hat{V}(\mathbf{s}) \right)^2 d\mathbf{s} \tag{3}$$

$$= \int_{\mathbf{s}} \mu(\mathbf{s}) \left( \int_{\mathbf{a}} \pi(\mathbf{a}|\mathbf{s}) \left( Q(\mathbf{s},\mathbf{a}) - \hat{V}(\mathbf{s}) \right) d\mathbf{a} \right)^2 d\mathbf{s}, \tag{4}$$

which is used to find an approximation $\hat{V}$ of the value function. $\mu(\mathbf{s})$ denotes the state distribution when following policy $\pi(\cdot|\mathbf{a})$. Since the squared function is convex we can use Jensens inequality for probability density functions to derive an upper bound of Equation (4)

$$\text{Error}(\hat{V}) \leq \int_{\mathbf{s}} \mu(\mathbf{s}) \int_{\mathbf{a}} \pi(\mathbf{a}|\mathbf{s}) \left( Q(\mathbf{s},\mathbf{a}) - \hat{V}(\mathbf{s}) \right)^2 d\mathbf{a}d\mathbf{s} = \text{Error}_B(\hat{V}). \tag{5}$$

The solution $\hat{V}^*$ for minimizing the upper bound $\text{Error}_B(\hat{V})$ is the same as for the original error function $\text{Error}(\hat{V})$.

*Proof.* To see this, we compute the square and replace the term $\int_{\mathbf{a}} \pi(\mathbf{a}|\mathbf{s})Q(\mathbf{s},\mathbf{a})d\mathbf{a}$ by the value function $V(\mathbf{s})$. This is done for the error function $\text{Error}(\hat{V})$ and for the upper bound $\text{Error}_B(\hat{V})$.

$$\text{Error}(\hat{V}) = \int_{\mathbf{s}} \mu(\mathbf{s}) \left( V(\mathbf{s}) - \hat{V}(\mathbf{s}) \right)^2 d\mathbf{s} = \int_{\mathbf{s}} \mu(\mathbf{s}) \left( V(\mathbf{s})^2 - 2V(\mathbf{s})\hat{V}(\mathbf{s}) + \hat{V}(\mathbf{s})^2 \right) d\mathbf{s} \tag{6}$$

$$\text{Error}_B(\hat{V}) = \int_{\mathbf{s}} \mu(\mathbf{s}) \int_{\mathbf{a}} \pi(\mathbf{a}|\mathbf{s}) \left( Q(\mathbf{s},\mathbf{a})^2 - 2Q(\mathbf{s},\mathbf{a})\hat{V}(\mathbf{s}) + \hat{V}(\mathbf{s})^2 \right) d\mathbf{a}d\mathbf{s} \tag{7}$$

$$= \int_{\mathbf{s}} \mu(\mathbf{s}) \left( \int_{\mathbf{a}} \pi(\mathbf{a}|\mathbf{s})Q(\mathbf{s},\mathbf{a})^2 d\mathbf{a} - 2V(\mathbf{s})\hat{V}(\mathbf{s}) + \hat{V}(\mathbf{s})^2 \right) d\mathbf{s} \tag{8}$$

Both error functions are the same except for an additive constant which does not depend on $\hat{V}$.

In difference to the original error function, the upper bound $\text{Error}_B$ can be approximated straightforwardly by samples $\{(\mathbf{s}_i, \mathbf{a}_i), Q(\mathbf{s}_i, \mathbf{a}_i)\}_{1 \leq i \leq N}$ gained by following some behavior policy $\pi_b(\cdot|\mathbf{s})$.

$$\text{Error}_B(\hat{V}) \approx \sum_{i=1}^{N} \frac{\mu(\mathbf{s})\pi(\mathbf{a}_i|\mathbf{s}_i)}{\mu_b(\mathbf{s}_i)\pi_b(\mathbf{a}_i|\mathbf{s}_i)} \left( Q(\mathbf{s}_i, \mathbf{a}_i) - \hat{V}(\mathbf{s}_i) \right)^2, \tag{9}$$

$\mu_b(\mathbf{s})$ defines the state distribution when following the behavior policy $\pi_b$. The term $1/(\mu_b(\mathbf{s}_i)\pi_b(\mathbf{s}_i, \mathbf{a}_i))$ ensures that we do not give more weight on states and actions preferred by $\pi_b$. This is a well known method in importance sampling. In order to keep our algorithm tractable, the factors $\pi_b(\mathbf{a}_i|\mathbf{s}_i)$, $\mu_b(\mathbf{s}_i)$ and $\mu(\mathbf{s}_i)$ will all be set to $1/N$. The minimization of Equation (9) defines a weighted regression problem which is given by the dataset $D_V$, the weighting $U$ and the weighted regression procedure WeightedRegress

$$D_V = \left\{ [(\mathbf{s}_i, \mathbf{a}_i), Q(\mathbf{s}_i, \mathbf{a}_i)]_{1 \leq i \leq N} \right\}, U = \left\{ [\pi(\mathbf{a}_i|\mathbf{s}_i)]_{1 \leq i \leq N} \right\}, \hat{V} = \text{WeightedRegress}(D_V, U) \tag{10}$$

**Algorithm 1** FQI with Advantage Weighted Regression

---

**Input:** $H = \{< \mathbf{s}_i, \mathbf{a}_i, r_i, \mathbf{s}'_i >\}_{1 \leq i \leq N}$, $\tau$ and $L$ (Number of Iterations)

Initialize $\hat{V}_0(\mathbf{s}) = 0$.

**for** $k = 0$ **to** $L-1$ **do**

$\quad D_k(\hat{V}_k) = \left\{ \left[ (\mathbf{s}_i, \mathbf{a}_i), r_i + \gamma \hat{V}_k(\mathbf{s}'_i) \right]_{1 \leq i \leq N} \right\}$

$\quad Q_{k+1} = \text{Regress}(D_k(\hat{V}_k))$

$\quad A(i) = Q_{k+1}(\mathbf{s}_i, \mathbf{a}_i) - \hat{V}_k(\mathbf{s}_i)$

$\quad$ Estimate $m_A(\mathbf{s}_i)$ and $\sigma_A(\mathbf{s}_i)$ for $1 \leq i \leq N$

$\quad U = \{[\exp(\tau(A(i) - m_A(\mathbf{s}_i))/\sigma_A(\mathbf{s}_i)]_{i \leq i \leq N}\}$

$\quad \hat{V}_{k+1} = \text{WeightedRegress}(D_k(\hat{V}_k), U)$

**end for**

---

The result shows that in order to approximate the value function $V(\mathbf{s})$, we do not need to carry out the expensive integration over the action space for each state $\mathbf{s}_i$. It is sufficient to know the Q-values at a finite set of state-action pairs.

## 3.2 Soft-greedy policy improvement

We use a soft-max policy [1] in the policy improvement step of the FQI algorithm. Our soft-max policy $\pi_1(\mathbf{a}|\mathbf{s})$ is based on the advantage function $A(\mathbf{s}, \mathbf{a}) = Q(\mathbf{s}, \mathbf{a}) - V(\mathbf{s})$. We additionally assume the knowledge of the mean $m_A(\mathbf{s})$ and the standard deviation of $\sigma_A(\mathbf{s})$ of the advantage function at state $\mathbf{s}$. These quantities can be estimated locally or approximated by additional regressions. The policy $\pi_1(\mathbf{a}|\mathbf{s})$ is defined as

$$\pi_1(\mathbf{a}|\mathbf{s}) = \frac{\exp(\tau \bar{A}(\mathbf{s}, \mathbf{a}))}{\int_{\mathbf{a}} \exp(\tau \bar{A}(\mathbf{s}, \mathbf{a})) d\mathbf{a}}, \quad \bar{A}(\mathbf{s}, \mathbf{a}) = \frac{A(\mathbf{s}, \mathbf{a}) - m_A(\mathbf{s})}{\sigma_A(\mathbf{s})}. \tag{11}$$

$\tau$ controls the greediness of the policy. If we assume that the advantages $A(\mathbf{s}, \mathbf{a})$ are distributed with $\mathcal{N}(A(\mathbf{s}, \mathbf{a})|m_A(\mathbf{s}), \sigma_A^2(\mathbf{s}))$, all normalized advantage values $\bar{A}(\mathbf{s}, \mathbf{a})$ have the same distribution. Thus, the denominator of $\pi_1$ is constant for all states and we can use the term $\exp(\tau \bar{A}(\mathbf{s}, \mathbf{a})) \propto \pi_1(\mathbf{a}|\mathbf{s})$ directly as weighting for the regression defined in Equation (10). The resulting approximated value function $\hat{V}(\mathbf{s})$ is used to replace the greedy operator $V(\mathbf{s}'_i) = \max_{\mathbf{a}'} Q(\mathbf{s}'_i, \mathbf{a}')$ in the FQI algorithm. The FQI by Advantage Weighted Regression (AWR) algorithm is given in Algorithm 1. As we can see, the Q-function $Q_k$ is only queried once for each step in the history $H$. Furthermore only already seen state action pairs $(\mathbf{s}_i, \mathbf{a}_i)$ are used for this query.

After the FQI algorithm is finished we still need to determine a policy for subsequent data collection. The policy can be obtained in the same way as for reward-weighted regression [13], only the advantage is used instead of the reward for the weighting - thus, we are optimizing the long term costs instead of the immediate one.

## 4 Locally-Advantage-WEighted Regression (LAWER)

Based on the FQI by AWR algorithm, we propose a new, computationally efficient fitted Q-iteration algorithm which uses Locally Weighted Regression (LWR, [14]) as function approximator. Similar to kernel based methods, our algorithm needs to be able to calculate the similarity $w_i(\mathbf{s})$ between a state $\mathbf{s}_i$ in the dataset $H$ and state $\mathbf{s}$. To simplify the notation, we will denote $w_i(\mathbf{s}_j)$ as $w_{ij}$ for all $\mathbf{s}_j \in H$. $w_i(\mathbf{s})$ is calculated by a Gaussian kernel $w_i(\mathbf{s}) = \exp(-(\mathbf{s}_i - \mathbf{s})^T \mathbf{D}(\mathbf{s}_i - \mathbf{s}))$. The diagonal matrix $\mathbf{D}$ determines the bandwidth of the kernel. Additionally, our algorithm also needs a similarity measure $w_{ij}^a$ between two actions $\mathbf{a}_i$ and $\mathbf{a}_j$. Again $w_{ij}^a$ can be calculated by a Gaussian kernel $w_{ij}^a = \exp(-(\mathbf{a}_i - \mathbf{a}_j)^T \mathbf{D}^a (\mathbf{a}_i - \mathbf{a}_j))$.

Using the state similarity $w_{ij}$, we can estimate the mean and the standard deviation of the advantage function for each state $\mathbf{s}_i$

$$m_A(\mathbf{s}_i) = \frac{\sum_j w_{ij} A(j)}{\sum_j w_{ij}}, \quad \sigma_A^2(\mathbf{s}_i) = \frac{\sum_j w_{ij}(A(j) - m_A(\mathbf{s}_j))^2}{\sum_j w_{ij}}. \tag{12}$$

## 4.1 Approximating the value functions

For the approximation of the Q-function, we use Locally Weighted Regression [14]. The Q-function is therefore given by:

$$Q_{k+1}(\mathbf{s}, \mathbf{a}) = \tilde{\mathbf{s}}_{\mathbf{A}}(\mathbf{S_A}^T \mathbf{W} \mathbf{S_A})^{-1} \mathbf{S_A}^T \mathbf{W} \mathbf{Q}_{k+1} \tag{13}$$

where $\tilde{\mathbf{s}}_{\mathbf{A}} = [1, \mathbf{s}^T, \mathbf{a}^T]^T$, $\mathbf{S_A} = [\tilde{\mathbf{s}}_{\mathbf{A}}(1), \tilde{\mathbf{s}}_{\mathbf{A}}(2), ..., \tilde{\mathbf{s}}_{\mathbf{A}}(N)]^T$ is the state-action matrix, $\mathbf{W} = \mathrm{diag}(w_i(\mathbf{s}) w_i^a(\mathbf{a}))$ is the local weighting matrix consisting of state and action similarities, and $\mathbf{Q}_{k+1} = [\tilde{Q}_{k+1}(1), \tilde{Q}_{k+1}(2), \ldots, \tilde{Q}_{k+1}(N)]^T$ is the vector of the Q-values (see Equation (1)).

For approximating the V-function we can multiplicatively combine the advantage-based weighting $u_i = \exp(\tau \bar{A}(\mathbf{s}_i, \mathbf{a}_i))$ and the state similarity weights $w_i(\mathbf{s})$. The value $V_{k+1}(\mathbf{s})$ is given by [1]:

$$V_{k+1}(\mathbf{s}) = \tilde{\mathbf{s}}(\mathbf{S}^T \mathbf{U} \mathbf{S})^{-1} \mathbf{S}^T \mathbf{U} \mathbf{Q}_{k+1}, \tag{14}$$

where $\tilde{\mathbf{s}} = [1, \mathbf{s}^T]^T$, $\mathbf{S} = [\tilde{\mathbf{s}}_1, \tilde{\mathbf{s}}_2, ..., \tilde{\mathbf{s}}_N]^T$ is the state matrix and $\mathbf{U} = \mathrm{diag}(w_i(\mathbf{s}) u_i)$ is the weight matrix. We bound the estimate of $\hat{V}_{k+1}(\mathbf{s})$ by $\max_{i|w_i(\mathbf{s})>0.001} Q_{k+1}(i)$ in order to prevent the local regression from adding a positive bias which might cause divergence of the value iteration.

A problem with nonparametric value function approximators is their strongly increasing computational complexity with an increasing number of data points. A simple solution to avoid this problem is to introduce a local forgetting mechanism. Whenever parts of the state space are oversampled, old examples in this area are removed from the dataset.

## 4.2 Approximating the policy

Similar to reward-weighted regression [13], we use a stochastic policy $\pi(\mathbf{a}|\mathbf{s}) = \mathcal{N}(\mathbf{a}|\mu(\mathbf{s}), \mathrm{diag}(\sigma^2(\mathbf{s})))$ with Gaussian exploration as approximation of the optimal policy. The mean $\mu(\mathbf{s})$ and the variance $\sigma^2(\mathbf{s})$ are given by

$$\mu(\mathbf{s}) = \tilde{\mathbf{s}}(\mathbf{S}^T \mathbf{U} \mathbf{S})^{-1} \mathbf{S}^T \mathbf{U} \mathbf{A}, \quad \sigma^2(\mathbf{s}) = \frac{\sigma_{\mathrm{init}}^2 \alpha_0 + \sum_i w_i(\mathbf{s}) u_i (\mathbf{a}_i - \mu(\mathbf{s}_i))^2}{\alpha_0 + \sum_i w_i(\mathbf{s}) u_i}, \tag{15}$$

where $\mathbf{A} = [\mathbf{a}_1, \mathbf{a}_2, \ldots, \mathbf{a}_N]^T$ denotes the action matrix. The variance $\sigma^2$ automatically adapts the exploration of the policy to the uncertainty of the optimal action. With $\sigma_{\mathrm{init}}^2$ and $\alpha_0$ we can set the initial exploration of the policy. $\sigma_{\mathrm{init}}$ is always set to the bandwidth of the action space. $\alpha_0$ sets the weight of the initial variance in comparision to the variance comming from the data, $\alpha_0$ is set to 3 for all experiments.

## 5 Evaluations

We evaluated the LAWER algorithm on three benchmark tasks, the pendulum swing up task, the acrobot swing up task and a dynamic version of the puddle-world (i.e., augmenting the puddle-world by velocities, inertia, etc.) with 2 and 3 dimensions. We compare our algorithm to tree-based FQI [8] (CE-Tree), neural FQI [6] (CE-Net) and LWR-based FQI (CE-LWR) which all use the Cross-Entropy (CE) optimization to find the maximum Q-values. For the CE optimization we used $n_{CE} = 10$ samples for one dimensional, $n_{CE} = 25$ samples for 2-dimensional and $n_{CE} = 64$ for 3-dimensional control variables. $e_{CE}$ was always set to $0.3 n_{CE}$ and we used $k_{CE} = 3$ iterations. To enforce exploration when collecting new data, a Gaussian noise of $\epsilon = \mathcal{N}(0, 1.0)$ was added to the CE-based policy. For the tree-based algorithm, an ensemble of $M = 20$ trees was used, $K$ was set to the number of state and action variables and $n_{min}$ was set to 2 (see [8]). For the CE-Net algorithm we used a neural network with 2 hidden layers and 10 neurons per layer and trained the network with the algorithm proposed in [6] for 600 epochs. For all experiments, a discount factor of $\gamma = 0.99$ was used. The immediate reward function was quadratic in the distance to the goal position $\mathbf{x}_G$ and in the applied torque/force $r = -c_1(\mathbf{x} - \mathbf{x_G})^2 - c_2 \mathbf{a}^2$. For evaluating the learning process, the exploration-free (i.e., $\sigma(\mathbf{s}) = 0$, $\epsilon = 0$) performance of the policy was evaluated after each data-collection/FQI cycle. This was done by determining the accumulated reward during an episode starting from the specified initial position. All errorbars represent a $95\%$ confidence interval.

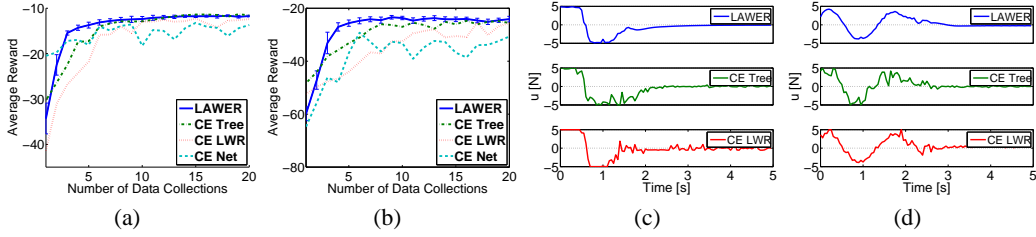

Figure 1: (a) Evaluation of LAWER and CE-based FQI algorithms on the pendulum swing-up task for $c_2 = 0.005$ . The plots are averaged over 10 trials. (b) The same evaluation for $c_2 = 0.025$. (c) Learned torque trajectories for $c_2 = 0.005$. (d) Learned torque trajectories for $c_2 = 0.025$.

## 5.1 Pendulum swing-up task

In this task, a pendulum needs to be swung up from the position at the bottom to the top position [6]. The state space consists of the angular deviation $\theta$ from the top position and the angular velocity $\dot{\theta}$ of the pendulum. The system dynamics are given by $0.5ml^2\ddot{\theta} = mg\sin(\theta) + u$ , the torque of the motor $u$ was limited to $[-5N, 5N]$. The mass was set to $m = 1$kg and length of the link to 1m. The time step was set to $0.05s$. Two experiments with different torque punishments $c_2 = 0.005$ and $c_2 = 0.025$ were performed.

We used $L = 150$ iterations. The matrices $\mathbf{D}$ and $\mathbf{D}_A$ were set to $\mathbf{D} = \text{diag}(30, 3)$ and $\mathbf{D}_A = \text{diag}(2)$. In the data collection phase, 5 episodes with 150 steps were collected starting from the bottom position and 5 episodes starting from a random position.

A comparison of the LAWER algorithm to CE-based algorithms for $c_2 = 0.005$ is shown in Figure 1(a) and for $c_2 = 0.025$ in Figure 1(b). Our algorithm shows a comparable performance to the tree-based FQI algorithm while being computationally much more efficient. All other CE-based FQI algorithms show a slightly decreased performance. In Figure 1(c) and (d) we can see typical examples of learned torque trajectories when starting from the bottom position for the LAWER, the CE-Tree and the CE-LWR algorithm. In Figure 1(c) the trajectories are shown for $c_2 = 0.005$ and in Figure 1(d) for $c_2 = 0.025$. All algorithms were able to discover a fast solution with 1 swing-up for the first setting and a more energy-efficient solution with 2 swing-ups for the second setting. Still, there are qualitative differences in the trajectories. Due to the advantage-weighted regression, LAWER was able to produce very smooth trajectories while the trajectories found by the CE-based methods look more jerky. In Figure 2(a) we can see the influence of the parameter $\tau$ on the performance of the LAWER algorithm. The algorithm works for a large range of $\tau$ values.

## 5.2 Acrobot swing-up task

In order to asses the performance of LAWER on a complex highly non-linear control task, we used the acrobot (for a description of the system, see [1]). The torque was limited to $[-5N, 5N]$. Both masses were set to 1kg and both lengths of the links to 0.5m. A time step of $0.1s$ was used. $L = 100$ iterations were used for the FQI algorithms. In the data-collection phase the agent could observe 25 episodes starting from the bottom position and 25 starting from a random position. Each episode had 100 steps. The matrices $\mathbf{D}$ and $\mathbf{D}_A$ were set to $\mathbf{D} = \text{diag}(20, 23.6, 10, 10.5)$ and $\mathbf{D}_A = \text{diag}(2)$. The comparison of the LAWER and the CE-Tree algorithm is shown in Figure 2(a). Due to the adaptive state discretization, the tree-based algorithm is able to learn faster, but in the end, the LAWER algorithm is able to produce policies of higher quality than the tree-based algorithm.

## 5.3 Dynamic puddle-world

In the puddle-world task [7], the agent has to find a way to a predefined goal area in a continuous-valued maze world (see Figure 3(a)). The agent gets negative reward when going through puddles. In difference to the standard puddle-world setting where the agent has a 2-dimensional state space (the $x$ and $y$ position), we use a more demanding setting. We have created a dynamic version of the puddle-world where the agent can set a force accelerating a $k$-dimensional point mass ($m = 1$kg).

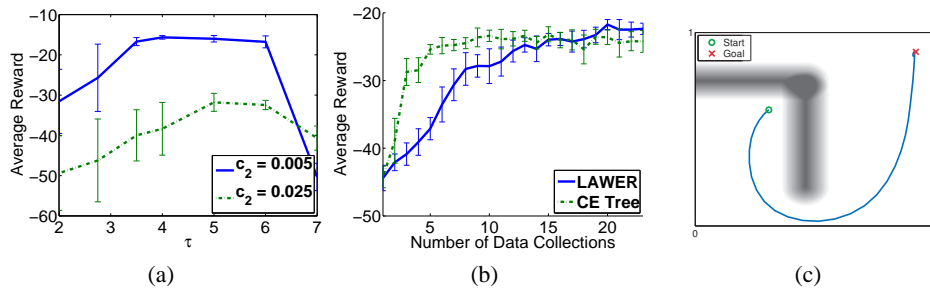

Figure 2: (a) Evaluation of the average reward gained over a whole learning trial on the pendulum swing-up task for different settings of $\tau$ (b) Comparison of the LAWER and the CE-Tree algorithm on the acrobot swing-up task (c) Setting of the 2-dimensional dynamic puddle-world.

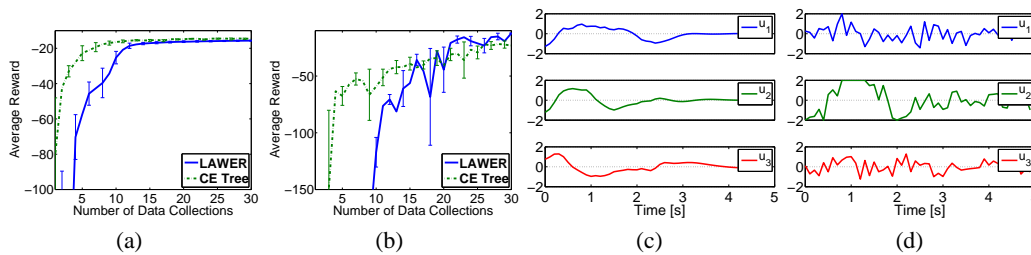

Figure 3: (a) Comparison of the CE-Tree and the LAWER algorithm for the 2-dimensional dynamic puddle-world. (b) Comparison of the CE-Tree and the LAWER algorithm for the 3-dimensional dynamic puddle-world. (c) Torque trajectories for the 3-dimensional puddle world learned with the LAWER algorithm. (d) Torque trajectories learned with the CE-Tree algorithm.

This was done for $k = 2$ and $k = 3$ dimensions. The puddle-world illustrates the scalability of the algorithms to multidimensional continuous action spaces (2 respectively 3 dimensional). The positions were limited to $[0, 1]$ and the velocities to $[-1, 1]$. The maximum force that could be applied in one direction was restricted to $2N$ and the time step was set to $0.1s$. The setting of the 2-dimensional puddle-world can be seen in Figure 2(c). Whenever the agent was about to leave the predefined area, the velocities were set to zero and an additional reward of $-5$ was given. We compared the LAWER with the CE-Tree algorithm. $L = 50$ iterations were used. The matrices $\mathbf{D}$ and $\mathbf{D}_A$ were set to $\mathbf{D} = \mathrm{diag}(10, 10, 2.5, 2.5)$ and $\mathbf{D}_A = \mathrm{diag}(2.5, 2.5)$ for the 2-dimensional and to $\mathbf{D} = \mathrm{diag}(8, 8, 8, 2, 2, 2)$ and $\mathbf{D}_A = \mathrm{diag}(1, 1, 1)$ for the 3-dimensional puddle-world. In the data collection phase the agent could observe 20 episodes with 50 steps starting from the predefined initial position and 20 episodes starting from a random position.

In Figure 3(a), we can see the comparison of the CE-Tree and the LAWER algorithm for the 2-dimensional puddle-world and in Figure 3(b) for the 3-dimensional puddle-world. The results show that the tree-based algorithm has an advantage in the beginning of the learning process. However, the CE-Tree algorithm has problems finding a good policy in the 3-dimensional action-space, while the LAWER algorithm still performs well in this setting. This can be seen clearly in the comparison of the learned force trajectories which are shown in Figure 3(c) for the LAWER algorithm and in Figure 3(d) for the CE-Tree algorithm. The trajectories for the CE-Tree algorithm are very jerky and almost random for the first and third dimension of the control variable, whereas the trajectories found by the LAWER algorithm look very smooth and goal directed.

## 6 Conclusion and future work

In this paper, we focused on solving RL problems with continuous action spaces with fitted Q-iteration based algorithms. The computational complexity of the max operator $\max_a Q(\mathbf{s}, \mathbf{a})$ often makes FQI algorithms intractable for high dimensional continuous action spaces. We proposed a

new method which circumvents the $\max$ operator by the use of a stochastic soft-max policy that allows us to reduce the policy improvement step $V(\mathbf{s}) = \max_a Q(\mathbf{s}, \mathbf{a})$ to a weighted regression problem. Based on this result, we can derive the LAWER algorithm, a new, computationally efficient FQI algorithm based on LWR.

Experiments have shown that the LAWER algorithm is able to produce high quality smooth policies, even for high dimensional action spaces where the use of expensive optimization methods for calculating $\max_a Q(\mathbf{s}, \mathbf{a})$ becomes problematic and only quite suboptimal policies are found. Moreover, the computational costs of using continuous actions for standard FQI are daunting. The LAWER algorithm needed on average 2780s for the pendulum, 17600s for the acrobot, 13700s for the 2D-puddle-world and 24200s for the 3D-puddle world benchmark task. The CE-Tree algorithm needed on average 59900s, 201900s, 134400s and 212000s, which is an order of magnitude slower than the LAWER algorithm. The CE-Net and CE-LWR algorithm showed comparable running times as the CE-Tree algorithm. A lot of work has been spent to optimize the implementations of the algorithms. The simulations were run on a P4 Xeon with 3.2 gigahertz.

Still, in comparison to the tree-based FQI approach, our algorithm has handicaps when dealing with high dimensional state spaces. The distance kernel matrices have to be chosen appropriately by the user. Additionally, the uniform distance measure throughout the state space is not adequate for many complex control tasks and might degrade the performance. Future research will concentrate on combining the AWR approach with the regression trees presented in [8].

# 7 Acknowledgement

This paper was partially funded by the Austrian Science Fund FWF project # P17229. The first author also wants to thank Bernhard Schölkopf and the MPI for Biological Cybernetics in Tübingen for the academic internship which made this work possible.

## Footnotes

[1] In practice, ridge regression $V_{k+1}(\mathbf{s}) = \tilde{\mathbf{s}}(\mathbf{S}^T \mathbf{W} \mathbf{S} + \sigma \mathbf{I})^{-1} \mathbf{S}^T \mathbf{W} \mathbf{Q}_{k+1}$ is used to avoid numerical instabilities in the regression.

# References

[1] R. Sutton and A. Barto, *Reinforcement Learning*. Boston, MA: MIT Press, 1998.

[2] J. A. Boyan and A. W. Moore, "Generalization in reinforcement learning: Safely approximating the value function," in *Advances in Neural Information Processing Systems 7*, pp. 369–376, MIT Press, 1995.

[3] P. Viviani and T. Flash, "Minimum-jerk, two-thirds power law, and isochrony: Converging approaches to movement planning," *Journal of Experimental Psychology: Human Perception and Performance*, vol. 21, no. 1, pp. 32–53, 1995.

[4] R. M. Alexander, "A minimum energy cost hypothesis for human arm trajectories," *Biological Cybernetics*, vol. 76, pp. 97–105, 1997.

[5] C. M. Harris and D. M. Wolpert, "Signal-dependent noise determines motor planning.," *Nature*, vol. 394, pp. 780–784, August 1998.

[6] M. Riedmiller, "Neural fitted Q-iteration - first experiences with a data efficient neural reinforcement learning method," in *Proceedings of the European Conference on Machine Learning (ECML)*, 2005.

[7] R. Sutton, "Generalization in reinforcement learning: Successful examples using sparse coarse coding," in *Advances in Neural Information Processing Systems 8*, pp. 1038–1044, MIT Press, 1996.

[8] D. Ernst, P. Geurts, and L. Wehenkel, "Tree-based batch mode reinforcement learning," *J. Mach. Learn. Res.*, vol. 6, pp. 503–556, 2005.

[9] A. Antos, R. Munos, and C. Szepesvari, "Fitted Q-iteration in continuous action-space MDPs," in *Advances in Neural Information Processing Systems 20*, pp. 9–16, Cambridge, MA: MIT Press, 2008.

[10] S. Timmer and M. Riedmiller, "Fitted Q-iteration with CMACs," pp. 1–8, 2007.

[11] J. Peters and S. Schaal, "Policy learning for motor skills," in *Proceedings of 14th International Conference on Neural Information Processing (ICONIP)*, 2007.

[12] P.-T. de Boer, D. Kroese, S. Mannor, and R. Rubinstein, "A tutorial on the cross-entropy method," *Annals of Operations Research*, vol. 134, pp. 19–67, January 2005.

[13] J. Peters and S. Schaal, "Reinforcement learning by reward-weighted regression for operational space control," in *Proceedings of the International Conference on Machine Learning (ICML)*, 2007.

[14] C. G. Atkeson, A. W. Moore, and S. Schaal, "Locally weighted learning," *Artificial Intelligence Review*, vol. 11, no. 1-5, pp. 11–73, 1997.
